# NEURAL NETWORKS THAT LEARN TO DISCRIMINATE SIMILAR KANJI CHARACTERS

Yoshihiro Mori          Kazuhiko Yokosawa
ATR Auditory and Visual Perception Research Laboratories
2-1-61 Shiromi Higashiku Osaka 540 Japan

## ABSTRACT

A neural network is applied to the problem of recognizing Kanji characters. Using a back propagation network learning algorithm, a three-layered, feed-forward network is trained to recognize similar handwritten Kanji characters. In addition, two new methods are utilized to make training effective. The recognition accuracy was higher than that of conventional methods. An analysis of connection weights showed that trained networks can discern the hierarchical structure of Kanji characters. This strategy of trained networks makes high recognition accuracy possible. Our results suggest that neural networks are very effective for Kanji character recognition.

## 1 INTRODUCTION

Neural networks are applied to recognition tasks in many fields, with good results. In the field of letter recognition, networks have been made which recognize hand-written digits [Burr 1986] and complex printed Chinese characters [Ho 1988]. The performance of these networks has been better than that of conventional methods. However, these results are still rudimentary when we consider not only the large number of Kanji characters, but the distortion involved in hand-written characters. We are aiming to make a large-scale network that recognizes the 3000 Kanji characters commonly used in Japan. Since it is difficult for a single network to discriminate 3000 characters, our plan is to create a large-scale network by

assembling many smaller ones that would each be responsible for recognizing only a    small number of characters.

There are two issues concerning implementation of such a large-scale network : the ability of individual networks, and organizing the networks. As a first step, the ability of a small network to discriminate similar Kanji characters was investigated. We found that the learning speed and performance of networks are highly influenced by environment (for instance, the order, number, and repetition of training samples). New methods of teaching the environment are utilized to make learning effective.

## 2 NEW TYPES OF TEACHERS

## 2.1 PROBLEMS OF BACKPROPAGATION

The Backpropagation(BP) learning algorithm only teaches correct answers [Rumelhart 1986]. BP does not care about the recognition rate of each category. If we use ordinary BP in a situation of limited resources, and if there are both easy and difficult categories to learn in the training set, what happens is that the easier category uses up most of the resources in the early stages of training (Figure 1). Yet, for efficiency, the difficult category to learn should get more resources. This weakness of BP makes the learning time longer.

Two new methods are used to avoid this problem. In the real world, learning procedures (human) do not exist in isolation. There is also a learning environment. It is therefore natural, and even necessary, to devise teaching methods that incorporate environmental    factors.

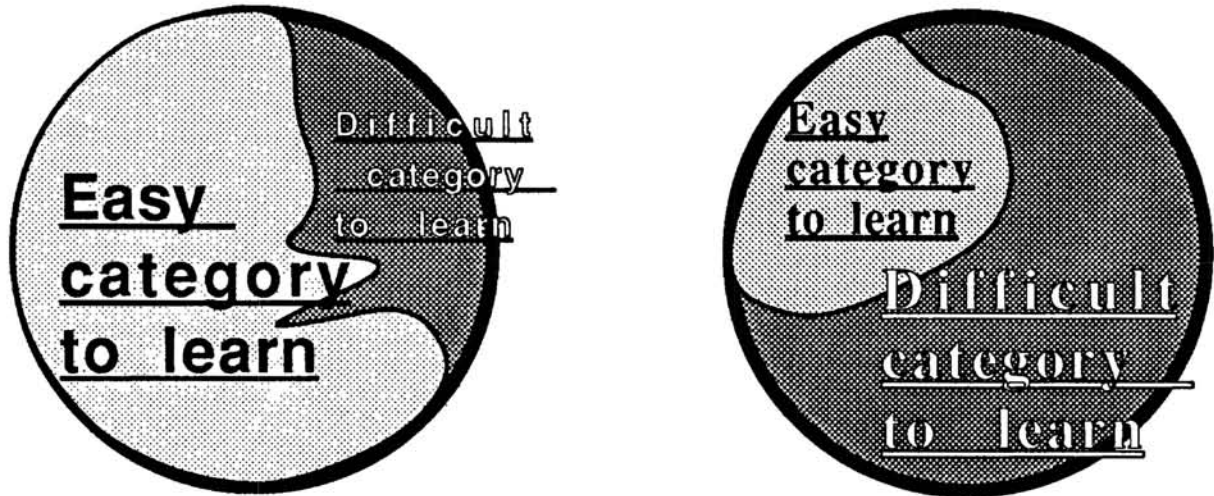

Separation by BP                    Ideal Separation
Figure 1. Easily Learned Category   Takes more Resources

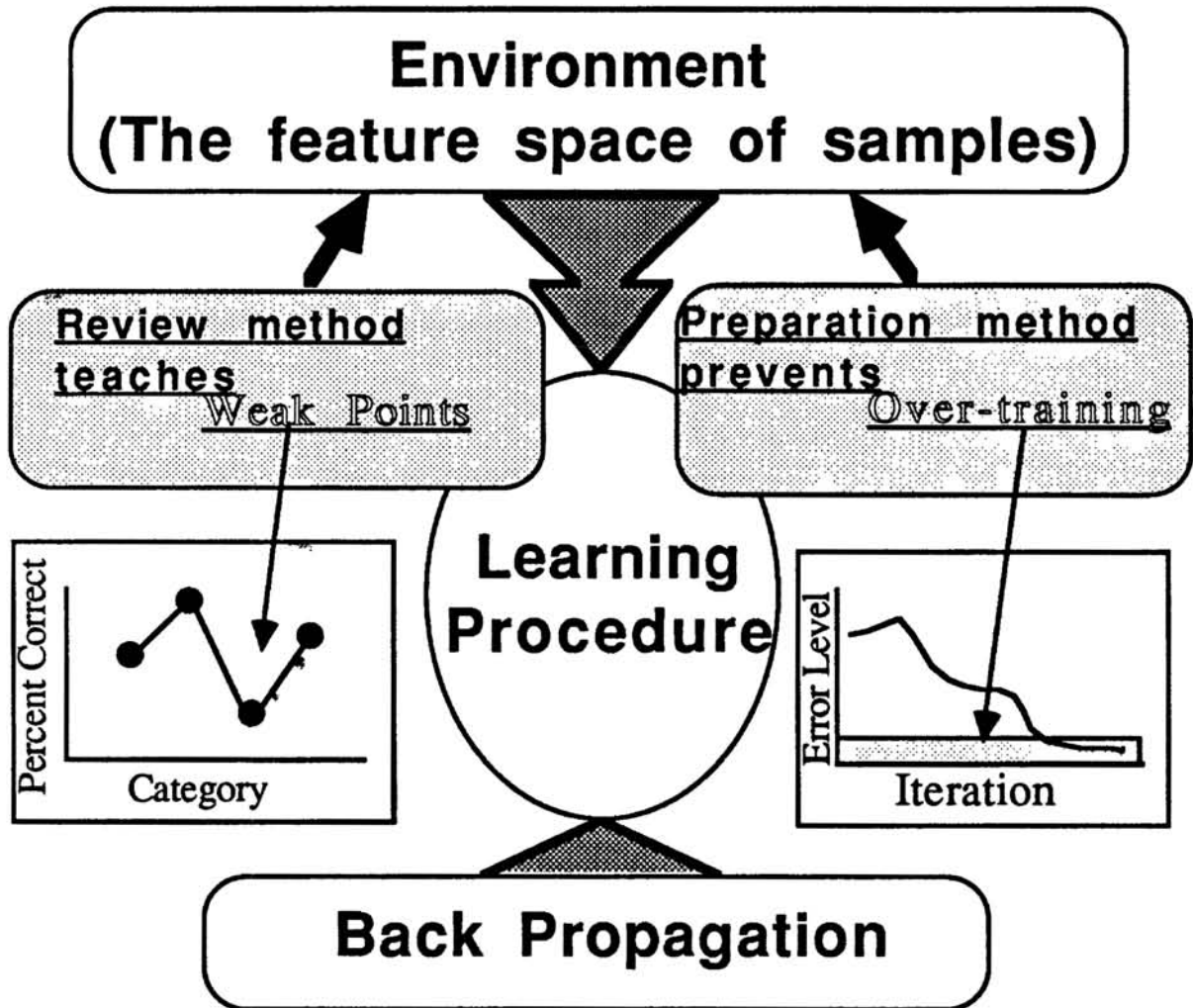

Figure 2.    Two New methods

## 2.2 FIRST METHOD (REVIEW METHOD)

This method tracks the performance for each category. At first, training is focused on categories that are not being recognized well. After this, on a more fine-grained level, the error for each sample is first checked, and the greater this error, the more often that sample is presented (Figure 2). This leads to a more balanced recognition over the categories.

## 2.3 SECOND METHOD (PREPARATION METHOD)

The second method, designed to prevent over-training, is to increase the number of training samples when the network's total error rate is observed to fall below a certain value (Figure 2).

# 3 RECOGNITION EXPERIMENT

## 3.1 INPUT PATTERN AND NETWORK STRUCTURE

Kanji characters are composed of sub-characters called radicals (Figure 3). The four Kanji characters used in our experiment are shown in Figure 4. These characters are all combinations of two kinds of left radicals and two kinds of right radicals. Visually, these characters are similar and hence difficult to discriminate. The training samples for this network were chosen from a database of about 3000 Kanji characters [Saito 1985]. For each character, there are 200 handwritten samples from different writers. 100 are used as training samples, and the remaining 100 are used to test recognition accuracy of the trained network. All samples in the database consist of 64 by 63 dots. If we were to use this pattern as the input to our neural net, the number of units required in the input layer would be too large for the computational abilities of current computers. Therefore, two kinds of feature vectors extracted from handwritten patterns are used as the input. In one of the feature vectors, the "MESH feature", there are 64 dimensions computing the density of the 8 by 8 small squares into which handwritten samples are divided. In the other, the "LDCD feature" [Hagita 1983], there are 256 dimensions computing a line segment length along four directions — horizontal, vertical, and two diagonals — in the same

small squares. In this experiment, we use a feed-forward neural network with three layers — an input layer, a hidden layer and an output layer —. Each unit of the input layer is connected to all units of the hidden layer, and each unit of the hidden layer is connected to all units of the output layer.

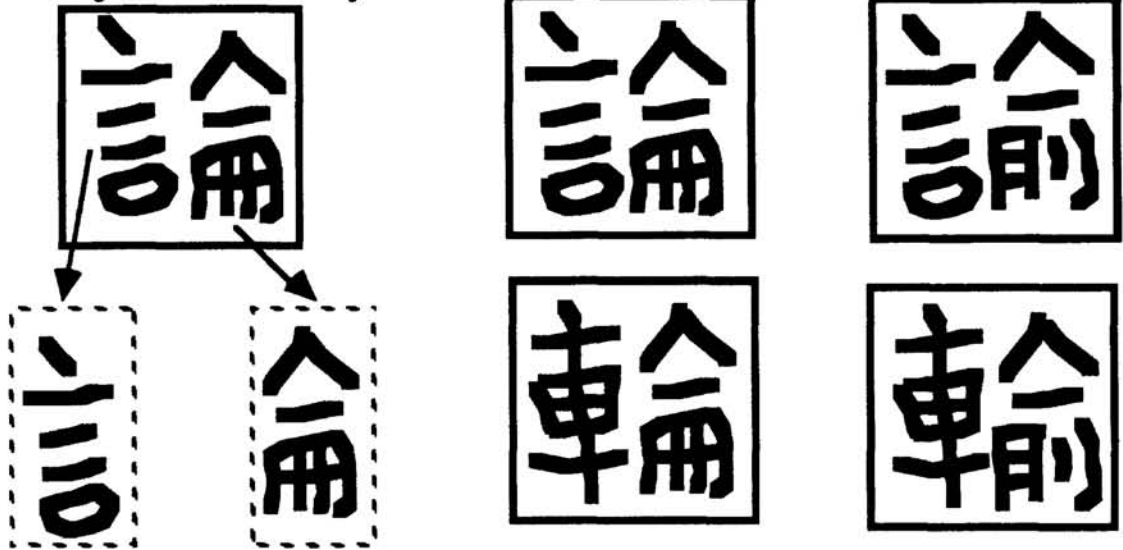

Fig. 3 Concept of Radical          Fig. 4    Example of Kanji Characters

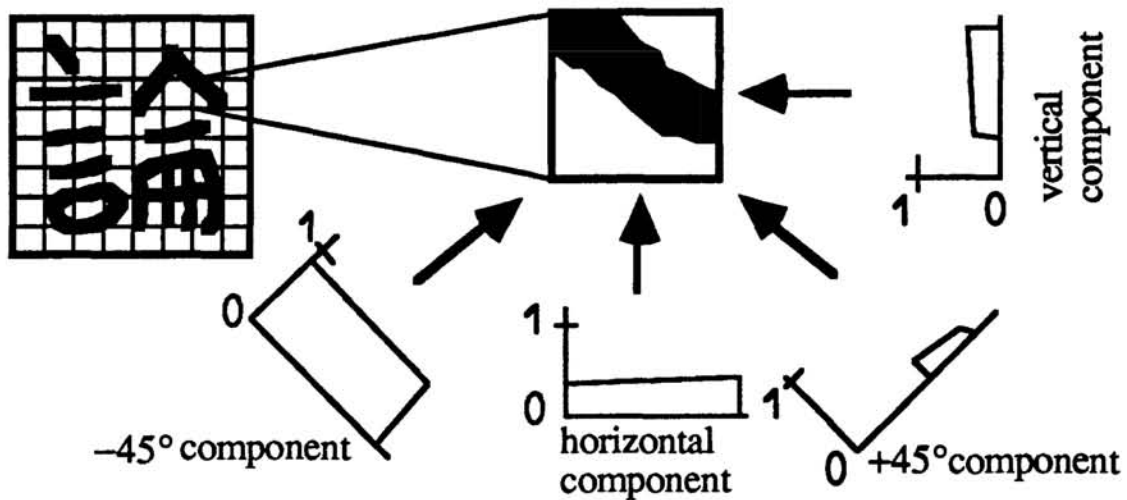

Figure 5.   LDCD Feature

## 3.2 RECOGNITION RESULTS (MESH VS. LDCD)

Average recognition rates when the MESH feature was used were 98.5% for training samples and 82.5% for testing samples. Average recognition rates when the LDCD feature was used were

99.5 % for training samples and 92.0% for testing samples. These recognition rates for neural networks were higher than for conventional methods we used.

## 3.3 Recognition Rate & the Number of Samples

We gradually increased the number of training samples to investigate the influence of this number on the recognition rate of testing samples. Figure 6 shows the recognition rate of testing samples for ten different amounts of training samples. When the number of training samples are 2 and 3, the recognition rates are lower than for 1 training sample. This result is probably due to the fact that the second samples in each set are not well-written. This result means that an average pattern should be used in the early training period.

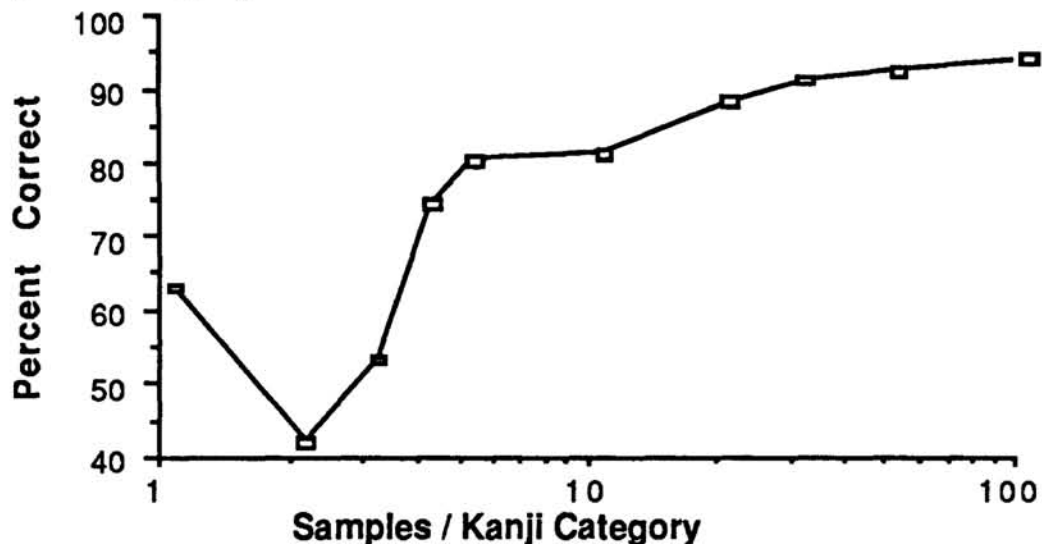

Figure 6.    Recognition Rate and the Number of Training Samples

## 3.4 ANALYSIS OF INNER REPRESENTATION

### 3.4.1 Weights vs. Difference Between Averaged Samples

To investigate how this neural network learns to solve the given task, the weights vector from the input layer to each hidden unit is compared to the difference between averaged samples with a common radical. Since the four Kanji characters in this task are all combinations of two kinds of left radicals and two kinds of right radicals, two hidden units which take charge of left and right radicals, respectively, are enough to accomplish

recognition. At first, 200 samples with the same left radical were averaged. Since there are just two left radicals in the four Kanji characters, this produced two averaged patterns. These two patterns were then subtracted, yielding a pattern that corresponds to the difference between the two left radicals. The same method was used to obtain a pattern that corresponds to the difference between the two right radicals. Then, for each of these patterns, the correlation coefficient with the weights from the input-layer to each hidden unit is calculated. The pattern for left radicals was very highly correlated with hidden unit 1 ($R=0.71, p<0.01$), and not correlated with hidden unit 2. On the other hand, the pattern for right radicals was very highly correlated with hidden unit 2 ($R=0.79, p<0.01$), and not correlated with hidden unit 1. In other words, each hidden unit is discriminating among radicals of one particular side of the Kanji characters.

### 3.4.2  Weights  vs.  Bayse  Discrimination

The bayse method is used as a discrimination function when the distribution of the categories is known. Supposing that the distribution of categories in this task is normal distribution and the covariance matrix of each category is equal, the discrimination function becomes first order as given below.

$$f(X) = (\mu l - \mu r)^t \Sigma X + c \qquad (1)$$

$\Sigma$     : Covariance Matrix with the same radical
$\mu l$     : Average Vector with the same left radical
$\mu r$     : Average Vector with the same right radical
$X$     : Input Feature Vector
$c$     : Constant

The input vector to the input layer is translated to a hidden unit as follows.

$$Y = WX + \theta \qquad (2)$$

$Y$     : Input Sum
$X$     : Input Feature  Vector
$W$     : Weights Matrix from Input Layer to a Hidden Unit
$\theta$     : Threshold

Equation (2) is similar to equation (1). If the network uses a strategy similar to bayse discrimination, there should be some

correlation between    bayse weights $(\mu l - \mu r)^t \Sigma$ in equation (1) and W in equation (2). When the correlation coefficient between bayse weights and the weights from the input layer to each hidden unit was calculated, there was no significant correlation between them (R=0.02,p>0.05). In other words, the network does not use a strategy like bayse discrimination.

# 4 CONCLUSION

For this experiment, we observed that the learning procedure is influenced by the surrounding environment. With this fact in mind, new methods were proposed to make training within a learning process more effective. These methods lead to balanced recognition rates over categories. The most important result from this experiment is that a network trained with BP can perceive that Kanji characters are composed of radicals. Based on this ability, it is possible to estimate the number of units required for the hidden-layer of a network. Such a network could then form the building block of a large-scale network capable of recognizing as many as the 3000 Kanji characters commonly used in Japan.

## Acknowledgments

We are grateful to Dr. Michio Umeda for his support and encouragement. Special thanks to Kazuki Joe for the ideas he provided in our many discussions and for his help in developing simulation programs.

## Reference

[Burr 1986]        D.J.Burr,"A Neural Network Digit Recognizer", IEEE-SMC,1621-1625,1986.
[Ho 1988]        A.Ho and W.Furmanski,"Pattern Recognition by Neural Network Model on Hypercubes",HCCA3-528
[Rumelhart 1986] D.E.Rumelhart et al,"Parallel Distributed Processing",vol.1,The MIT Press,1986.
[Saito 1985]        T.Saito,H.Yamada,K.Yamamoto,"On the Data Base ETL9 of Handprinted Characters in JIS Chinese Characters and Its Analysis",J68-D,4,757-764,1985
[Hagita 1983]        N.Hagita,S.Naito,I.Masuda,"Recognition of Handprinted Chinese Characters by Global and Local Direction Contributivity Density-Feature", J66-D,6,722-729,1983